# NEW HARDWARE FOR MASSIVE NEURAL NETWORKS

D. D. Coon and A. G. U. Perera
Applied Technology Laboratory
University of Pittsburgh
Pittsburgh, PA 15260.

## ABSTRACT

Transient phenomena associated with forward biased silicon $p^+ - n - n^+$ structures at 4.2K show remarkable similarities with biological neurons. The devices play a role similar to the two-terminal switching elements in Hodgkin-Huxley equivalent circuit diagrams. The devices provide simpler and more realistic neuron emulation than transistors or op-amps. They have such low power and current requirements that they could be used in massive neural networks. Some observed properties of simple circuits containing the devices include action potentials, refractory periods, threshold behavior, excitation, inhibition, summation over synaptic inputs, synaptic weights, temporal integration, memory, network connectivity modification based on experience, pacemaker activity, firing thresholds, coupling to sensors with graded signal outputs and the dependence of firing rate on input current. Transfer functions for simple artificial neurons with spiketrain inputs and spiketrain outputs have been measured and correlated with input coupling.

## INTRODUCTION

Here we discuss the simulation of neuron phenomena by electronic processes in silicon from the point of view of hardware for new approaches to electronic processing of information which parallel the means by which information is processed in intelligent organisms. Development of this hardware basis is pursued through exploratory work on circuits which exhibit some basic features of biological neural networks. Fig. 1 shows the basic circuit used to obtain spiketrain outputs. A distinguishing feature of this hardware basis is the spontaneous generation of action potentials as a device physics feature.

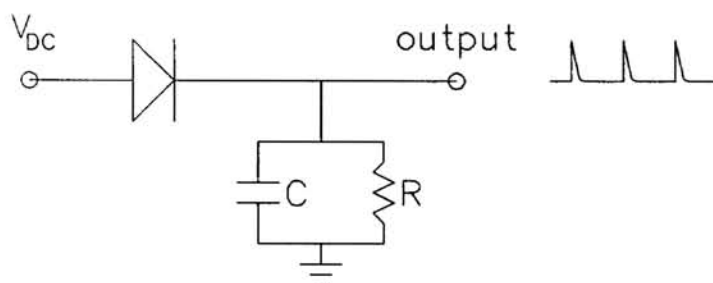

Figure 1: Spontaneous, neuronlike spiketrain generating circuit. The spikes are nearly equal in amplitude so that information is contained in the frequency and temporal pattern of the spiketrain generation.

## TWO-TERMINAL SWITCHING ELEMENTS

The use of transistor based circuitry[1] is avoided because transistor electrical characteristics are not similar to neuron characteristics. The use of devices with fundamentally non-neuronlike character increases the complexity of artificial neural networks. Complexity would be an important drawback for massive neural networks and most neural networks in nature achieve their remarkable performance through their massive size. In addition, transistors have three terminals whereas the switching elements of Hodgkin-Huxley equivalent circuits have two terminals. Motivated in part by Hodgkin-Huxley equivalent circuit diagrams, we employ two-terminal $p^+ - n - n^+$ devices which execute transient switching between low conductance and high conductance states. (See Fig. 2) We call these devices injection mode devices (IMDs). In the "OFF-STATE", a typical current through the devices is $\sim 100\,\text{fA}/\text{mm}^2$, and in the "ON-STATE" a typical current is $\sim 10\,\text{mA}/\text{mm}^2$. Hence this device is an extremely good switch with a ON/OFF ratio of $10^{11}$. As in real neurons[2], the current in the device is a function of voltage and time, not only voltage. The devices require cryogenic cooling but this results in an advantageously low quiescent power drain of $< 1\,\text{nanowatt}/\text{cm}^2$ of chip area and the very low leakage currents mentioned above. In addition, the highly unique ability of the neural networks described here to operate in a cryogenic environment is an important advantage for infrared image processing at the focal plane (see Fig. 3 and further discussion below). Vision systems begin processing at the focal plane and there are many benefits to be gained from the vision system approach to IR image processing.

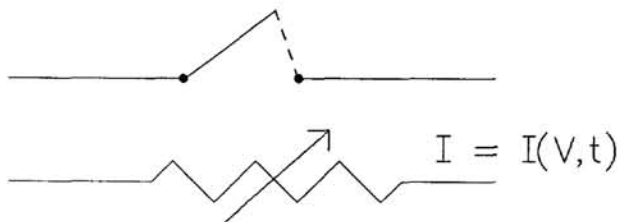

Figure 2: Switching element in Hodgkin-Huxley equivalent circuits.

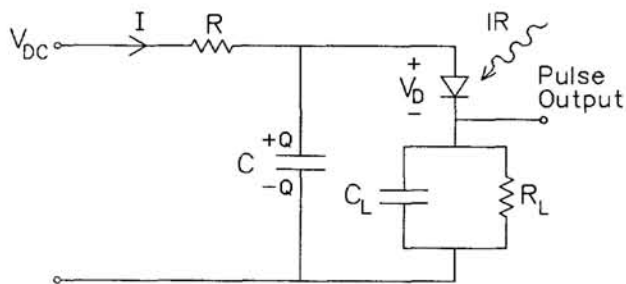

Figure 3: Single stage conversion of infrared intensity to spiketrain frequency with a neuron-like semiconductor device. No pre-amplifiers are necessary.

Coding of graded input signals (see Fig. 4) such as photocurrents into action potential spike trains with millimeter scale devices has been experimentally demonstrated[3] with currents from $1\,\mu A$ down to about 1 picoampere with coding noise referred to input of $< 10$ femtoamperes. Coding of much smaller current levels should be possible with smaller devices. Figure 5 clearly shows the threshold behavior of the IMD. For devices studied to date, a transition from action potential output to graded signal output is observed for input currents of the order of 0.5 picoamperes[13]

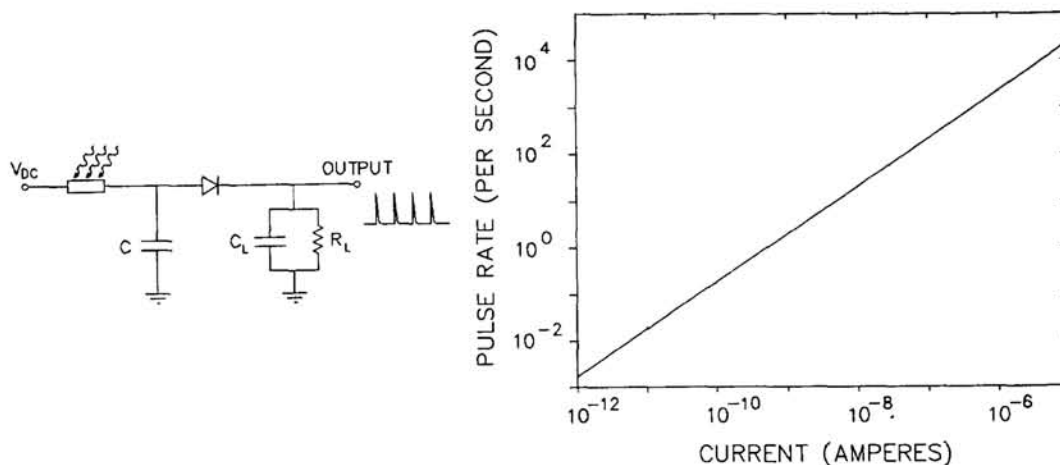

Figure 4: Coding of NIR-VISIBLE-UV intensity into firing frequency of a spiketrain and the experimentally determined firing rate vs. the input current for one device. Note that the dynamic range is about $10^7$.

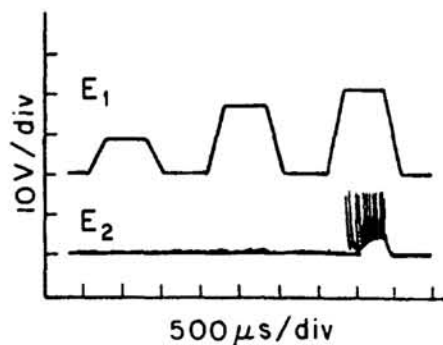

Figure 5: Illustration of the threshold firing of the device in response to input step functions.

This transition is remarkably well described in von Neumann's discussion[5,6] of the mixed character of neural elements which he relates to the concept of subliminal stimulation levels which are too low to produce the stereotypical all-or-nothing response. Neural network modelers frequently adopt viewpoints which ignore this interesting mixed character. The von Neumann viewpoint links the mixed character to concepts of nonlinear dynamics in a way which is not apparent in recent neural network modeling literature. The scaling down of IMD size should result in even lower current requirements for all-or-nothing response.

## DEVICE PHYSICS

Recently, neuronlike action potential transients in IMDs have been the subject of considerable research[3,4,7,8,9,10,11,12,13]. In the simple circuits of Fig. 1, the IMD gives rise to a spontaneous neuronlike spiketrain output. Between pulses, the IMD is polarized in the sense that it is in a low conductance state with a substantial voltage occurring across it, even though it is forward biased. The low conductance has been attributed to small interfacial work functions due to band offsets at the $n^+$-n and $p^+$-n interfaces[8].

Low temperatures inhibit thermionic injection of electrons and holes into the n-region from the $n^+$-layer and $p^+$-layer impurity bands[14]. Pulses are caused by

switching to depolarized states with low diode potential drops and large injection currents which are believed to be triggered by the slow buildup of a small thermionic injection current from the n$^+$-layer into the n-region. The injection current can cause impact ionization of n-region donor impurities resulting in an increasingly positive space charge which further enhances the injection current to the point where the IMD abruptly switches to the low conductance state with large injection current. Switching times are typically under 100ns. Charging of the load capacitance $C_L$ cuts off the large injection current and resets the diode to its low conductance state. The load capacitor $C_L$ then discharges through $R_L$. During the $C_L$ discharging time constant $R_L C_L$ the voltage across the IMD itself is low and therefore the bias voltage would have to be raised substantially to cause further firing. Thus, $R_L C_L$ is analogous to the refractory period of a neuron. The output pulses of an IMD generally have about the same amplitude while the rate of pulsing varies over a wide range depending on the bias voltage and the presence of electromagnetic radiation.[7,8,10]

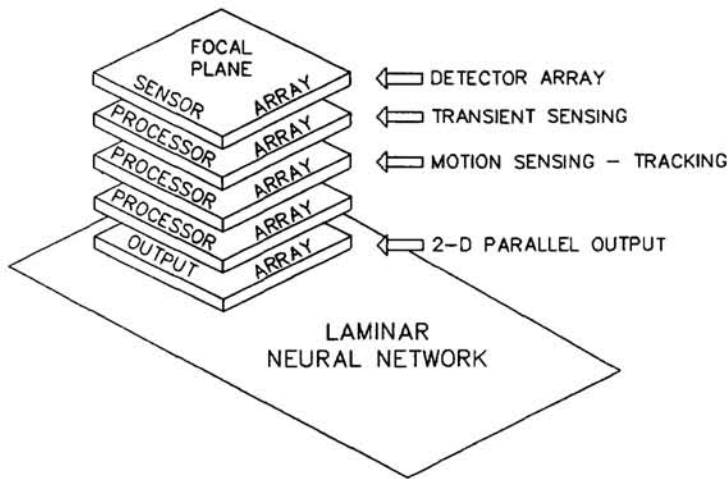

Figure 6: Illustrative laminar architecture showing stacked wafers in 3-dimensions.

## REAL TIME PARALLEL ASYNCHRONOUS PROCESSING

The devices described here could form the hardware basis for a parallel asynchronous processor in much the same way that transistors form the basis for digital computers. The devices could be used to construct networks which could perform real time signal processing. Pulse propagation through silicon chips (parallel firethrough, see Fig. 7) as opposed to the lateral planar propagation in conventional integrated circuits has been proposed.[15] This would permit the use of laminar, stacked wafer architectures. See Fig. 6.

Such architectures would eliminate the serial processing limitations of standard processors which utilize multiplexing and charge transfer. There are additional advantages in terms of elimination of pre-amplifiers and reduction in power consumption. The approach would utilize the *low power, low noise* devices[10] described here to perform input signal-to-frequency conversion in every processing channel.

## POWER CONSUMPTION FOR A BRAIN SCALE SYSTEM

The low power and low current requirements together with the electronic simplicity (lower parts-count as compared with transistor and op-amp approaches) and

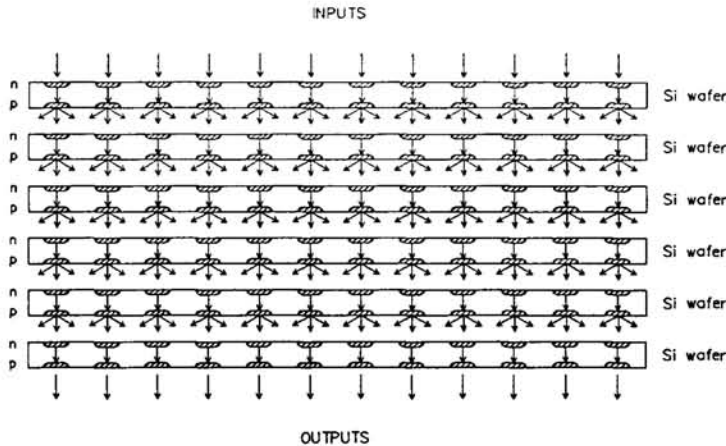

OUTPUTS

Figure 7: Schematic illustration of the signal flow pattern through a real time parallel asynchronous processor consisting of stacked silicon wafers.

the natural emulation of neuron features means that the approach described here would be especially advantageous for very large neural networks, e.g. systems comparable to supercomputers in which power dissipation and system complexity are important considerations. The *power consumption* of large scale analog[16] and digital[17] systems is always a major concern. For example, the power consumption of the CRAY XMP-48 is of the order of 300 kilowatts. For the devices described here, the power consumption is very low. For these devices, we have observed quiescent power drains of about 1 nW/cm$^2$ and pulse power consumption of about 500 nJ/pulse/cm$^2$. We estimate that a system with $10^{11}$ active $10\,\mu m \times 10\,\mu m$ elements (comparable to the number of neurons in the brain[18]) all firing with an average pulse rate of 1 KHz (corresponding to a high neuronal firing rate[5]) would consume about 50 watts. The quiescent power drain for this system would be 0.1 milliwatts. Thus, power (P) requirements for such an artificial neural network with the size scale ($10^{11}$ pulse generating elements) of the human brain and a range of activity between zero and the maximum conceivable sustained activity for neurons in the brain would be 0.1 milliwatts < P < 50 watts for 10 micron technology. For comparison, we note that von Neumann's estimate for the power dissipation of the brain is of order 10 to 25 watts.[5,6] Fabrication of a $10^{11}$ element $10\,\mu m$ artificial neural network would require processing of about 1500 four inch wafers.

## NETWORK CONNECTIVITY

For a network with coupling between many IMD's[3] we have shown[4] that

$$C_i R_i \frac{dV_i}{dt} + V_i = \sum_{j=1}^{N} T_{ij} F_j(V_j) + R_i I_i \tag{1}$$

where $V_i$ is the voltage across the diode and the input capacitance $C_i$ of the i-th network node, $R_i$ represents a leakage resistance in parallel with $C_i$, and $I_i$ represents an external current input to the i-th diode. i,j=1,2,3,..... label different network nodes and $T_{ij}$ incoporates coupling between network elements. Equation 1 has the same form as equations which occur in the Hopfield model[20,21,22,23] for neural networks. Sejnowski has also discussed similar equations in connection with skeleton filters in

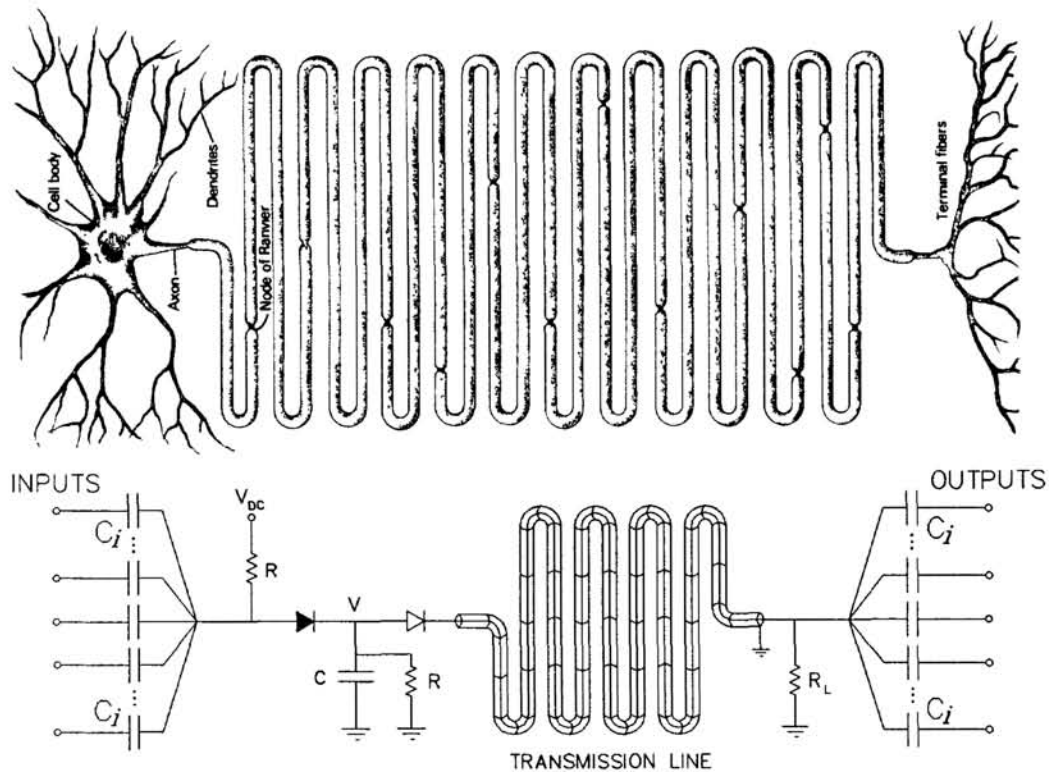

Figure 8: a) Main features of a typical neuron from Kandel and Schwartz.[19] b) Our artificial neuron, which shows the summation over synaptic inputs and fan-out.

the brain.[24,25] Nonlinear threshold behavior of IMD's enters through $F(V)$ as it does in the neural network models.

In Fig. 8-b a range of input capacitances is possible. This range of capacitances is related to the range of possible synaptic weights. The circuit in Fig. 8 accomplishes pulse height discrimination and each pulse can contribute to the charge stored on the central node capacitance $C$. The charge added to $C$ during each input pulse is linearly related to the input capacitance except at extreme limits. The range of input capacitances for a particular experiment was $.002\,\mu F$ to $.2\,\mu F$ which differ by a factor of about 100. The effect of various input capacitance values (synaptic weights) on input-output firing rates is shown in Fig. 9. Also the Fig. 8-b shows many capacitive inputs/outputs to/from a single IMD. i.e. fan-in and fan-out. For pulses which arrive at different inputs at about the same time, the effect of the pulses is additive. The time within which inputs are summed is just the stored charge lifetime. Summation over many inputs is an important feature of neural information processing.

## EXCITATION, INHIBITION, MEMORY

Both excitatory and inhibitory input circuits are shown in Fig. 10. Input pulses cause the accumulation of charge on $C$ in excitatory circuits and the depletion of charge on $C$ in inhibitory circuits. Charge associated with input spiketrains is integrated/stored on $C$. The temporally integrated charge is depleted by the firing of the IMD. Thus, the storage time is related to the firing rate. After an input spiketrain raises the potential across $C$ to a value above the firing threshold, the resulting IMD

statistically

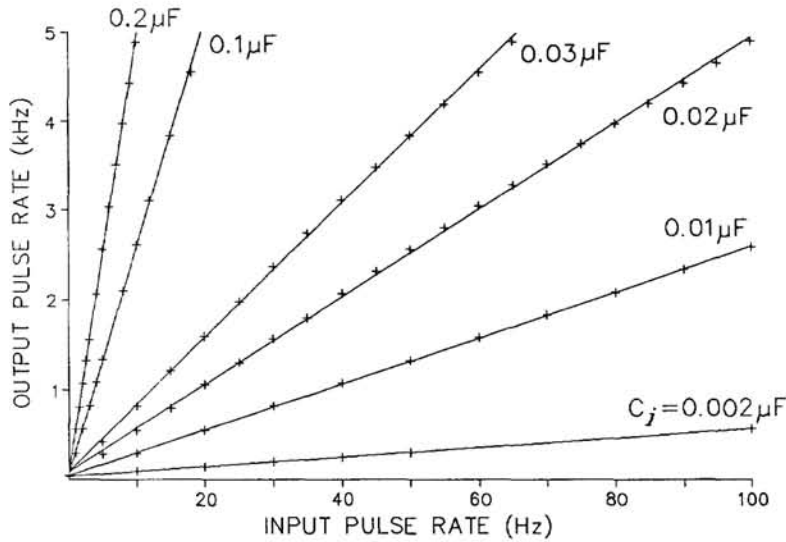

Figure 9: Output pulse rate vs. the input pulse rate for different input capacitance values $C_i$ values

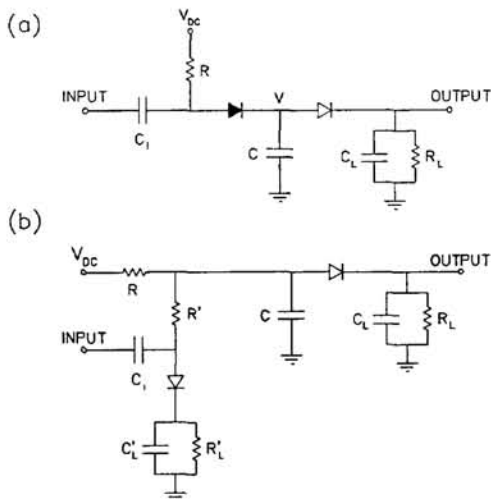

Figure 10: Circuits which incorporate rectifying synaptic inputs. a) an excitatory input. b) an inhibitory input.

output spiketrain codes the input information. The output firing rate is linearly related to the input firing rate times the synaptic coupling strength (linearly related to $C_i$). See Fig. 9. If the input ceases, then the potential across C relaxes back to a value just below the firing threshold. When not firing, the IMD has a high impedance. If there is negligible leakage of charge from C, then V can remain near $V_T$ (threshold voltage) for a long time and a new input signal will quickly take the IMD over the firing threshold. See Fig. 11. We have observed stored charge lifetimes of 56 days and longer times may be acheivable. The lifetime of charge stored on C can be reduced by adding a resistance in parallel with C.

From the discussion of integration, we see that long term storage of charge on C is equivalent to long term memory. The memory can be read by seeing if a new input pulse or spiketrain produces a prompt output pulse or spiketrain. The read signal input channel in Fig. 8-b can be the same as or different from the channel which resulted in the charge storage. In either case memory would produce a change in the pattern of connectivity if the circuit was imbedded in a neural network. Changes in patterns of connectivity are similar to Hebb's rule considerations[26] in which memory is associated with increases in the strength (weight) of synaptic couplings. Frequently,

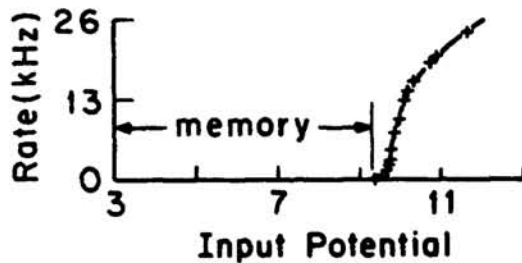

Figure 11: Firing rate vs. the bias voltage. The region where the firing is negligible is associated with memory. The state of the memory is associated with the proximity to the firing threshold.

the increase in synaptic weights is modeled by increased conductance whereas in the circuits in Figs. 10(a) and 8-b memory is achieved by integration and charge storage. Note that for these particular circuits, the memory is not eraseable although volatile (short term) memory can easily be constructed by adding a resistor in parallel with C. Thus, a continuous range of memory lifetimes can be achieved.

## 2-D PARALLEL ASYNCHRONOUS CHIP-TO-CHIP TRANSMISSION

For many IMD's the output pulse heights for a circuit like that in Fig. 1 are >3 volts. Thus, output from the first stage or any later stage of the network could easily be transmitted to other parts of an overall system. Two-dimensional arrays of devices on different chips could be coupled by indium bump bonding to form the laminar architecture described above. Planar technology could be used for local lateral interconnections in the processor. (See Fig. 7) In addition to transmission of electrical pulses, optical transmission is possible because the pulses can directly drive LED's.

Emerging GaAs-on-Si technology is interesting as a means of fabricating two dimensional emitter arrays. Optical transmission is not necessary but it might be useful (A) for processed image data transfer, (B) for coupling to an optical processor, or (C) to provide 2-D optical interconnects between chips bearing 2-D arrays of $p^+ - n - n^+$ diodes. Note that with optical interconnects between chips, the circuits employed here would be internal receivers. The p-i-n diodes employed in the present work would be well suited to the receiver role. An interesting possibility would entail the use optical interconnects between chips to achieve local, lateral interaction. This would be accomplished by having each optical emitter in a 2-D array broadcast locally to multiple receivers rather than to a single receiver. Similarly, each receiver would have a receptive field extending over multiple transmitters. It is also possible that an optical element could be placed in the gap between parallel transmitter and receiver planes to structure, control or alter 2-D patterns of interconnection. This would be an alternative to a planar technology approach to lateral interconnection. If the optical elements were active then the system would constitute a hybrid optical/electronic processor, whereas if passive optical elements were employed, we would regard the system as an optoelectronic processor. In either case, we picture the processing functions of temporal integration, spatial summation over inputs, coding and pulse generation as residing on-chip.

## ACKNOWLEDGEMENTS

The work was supported in part by U.S. DOE under contract #DE-ACO2-80ER10667 and NSF under grant # ECS-8603075.

# References

[1] L. D. Harmon, Kybernetik **1**, 89 (1961).

[2] A. L. Hodgkin and A. F. Huxley, J. Physiol **117**, 500 (1952).

[3] D. D. Coon and A. G. U. Perera, Int. J. Electronics **63**, 61 (1987).

[4] K. M. S. V. Bandara, D. D. Coon and R. P. G. Karunasiri, *Infrared Transient Sensing*, to be published.

[5] J. von Neumann, *The Computer and the Brain*, Yale University Press, New Haven and London, 1958.

[6] J. von Neumann, *Collected Works*, Pergamon Press, New York, 1961.

[7] D. D. Coon and A. G. U. Perera, Int. J. Infrared and Millimeter Waves **7**, 1571 (1986).

[8] D. D. Coon and S. D. Gunapala, J. Appl. Phys **57**, 5525 (1985).

[9] D. D. Coon, S. N. Ma and A. G. U. Perera, Phys. Rev. Let. **58**, 1139 (1987).

[10] D. D. Coon and A. G. U. Perera, Applied Physics Letters **51**, 1711 (1987).

[11] D. D. Coon and A. G. U. Perera, Solid-State Electronics **29**, 929 (1986).

[12] D. D. Coon and A. G. U. Perera, Applied Physics Letters **51**, 1086 (1987).

[13] K. M. S. V. Bandara, D.D. Coon and R. P. G. Karunasiri, Appl. Phys. Lett **51**, 961 (1987).

[14] Y. N. Yang, D. D. Coon and P. F. Shepard, Applied Physics Letters **45**, 752 (1984).

[15] D. D. Coon and A. G. U. Perera, Int. J. IR and Millimeter Waves **8**, 1037 (1987).

[16] M. A. Sivilotti, M. R. Emerling and C. A. Mead, *VLSI Architectures for Implementation of Neural Networks*, **Neural Networks for Computing**, A.I.P., 1986, pp. 408–413.

[17] R. W. Keyes, Proc. IEEE **63**, 740 (1975).

[18] E. R. Kandel and J. H. Schwartz, *Principles of Neural Science*, Elsevier, New York, 1985.

[19] E. R. Kandel and J. H. Schwartz, *Principles of Neural Science*, Elsevier, New York, 1985, page 15, Reproduced by permission of Elsevier Science Publishing Co., N.Y..

[20] J. J. Hopfield, Proc. Natl. Acad. Sci. U.S.A **81**, 3088 (1984).

[21] J. J. Hopfield and D. W. Tank, Biol. Cybern **52**, 141 (1985).

[22] J. J. Hopfield and D. W. Tank, Science **233**, 625 (1986).

[23] D. W. Tank and J. J. Hopfield, IEEE. Circuits Syst. **CAS-33**, 533 (1986).

[24] T. J. Sejnowski, J. Math. Biology **4**, 303 (1977).

[25] T. J. Sejnowski, *Skeleton Filters in the Brain*, Lawrence Erlbaum, New Jersey, 1981, pp. 189–212, edited by G. E. Hinton and J. A. Anderson.

[26] J. L. McClelland, D. E. Rumelhart and the PDP research group, *Parallel Distributed Processing*, The MIT Press, Cambridge, Massachusetts, 1986, two volumes.
